# On the Non-Existence of a Universal Learning Algorithm for Recurrent Neural Networks

**Herbert Wiklicky**
Centrum voor Wiskunde en Informatica
P.O.Box 4079, NL-1009 AB Amsterdam, The Netherlands
e-mail: herbert@cwi.nl

## Abstract

We prove that the so called "loading problem" for (recurrent) neural networks is *unsolvable*. This extends several results which already demonstrated that training and related design problems for neural networks are (at least) NP-complete. Our result also implies that it is impossible to find or to formulate a universal training algorithm, which for any neural network architecture could determine a correct set of weights. For the simple proof of this, we will just show that the loading problem is equivalent to "Hilbert's tenth problem" which is known to be unsolvable.

## 1 THE NEURAL NETWORK MODEL

It seems that there are relatively few commonly accepted general formal definitions of the notion of a "neural network". Although our results also hold if based on other formal definitions we will try to stay here very close to the original setting in which Judd's NP completeness result was given [Judd, 1990]. But in contrast to [Judd, 1990] we will deal here with simple recurrent networks instead of feed forward architectures.

Our networks are constructed from three different types of units: $\Sigma$-**units** compute just the sum of all incoming signals; for $\Pi$-**units** the activation (node) function is given by the product of the incoming signals; and with $\Theta$-**units** – depending if the input signal is smaller or larger than a certain threshold parameter $\theta$ – the output is zero or one. Our units are connected or linked by real weighted connections and operate synchronously.

Note that we could base our construction also just on one general type of units, namely what usually is called $\Sigma\Pi$-**units**. Furthermore, one could replace the $\Pi$-units in the below

construction by (recurrent) modules of simple linear threshold units which had to perform unary integer multiplication. Thus, no higher order elements are actually needed.

As we deal with recurrent networks, the behavior of a network now is not just given by a simple mapping from input space to output space (as with feed forward architectures). In general, an input pattern now is mapped to an (infinite) output sequence. But note, that if we consider as the output of a recurrent network a certain final, stable output pattern, we could return to a more static setting.

## 2   THE MAIN RESULT

The question we will look at is how difficult it is to construct or train a neural network of the described type so that it actually exhibits a certain desired behavior, i.e. solves a given learning task. We will investigate this by the following decision problem:

**Decision 1 Loading Problem**
*INSTANCE: A neural network architecture $N$ and a learning task $T$.*
*QUESTION: Is there a configuration $C$ for $N$ such that $T$ is realized by $C$?*

By a network configuration we just think of a certain setting of the weights in a neural network. Our main result concerning this problem now just states that it is undecidable or unsolvable.

**Theorem 1** *There exists no algorithm which could decide for any learning task $T$ and any (recurrent) neural network (consisting of $\Sigma$-, $\Pi$-, and $\Theta$-units) if the given architecture can perform $T$.*

The decision problem (as usual) gives a "lower bound" on the hardness of the related constructive problem [Garey and Johnson, 1979]. If we could *construct* a correct configuration for all instances, it would be trivial to *decide* instantly if a correct configuration exists at all. Thus we have:

**Corollary 2** *There exists no universal learning algorithm for (recurrent) neural networks.*

## 3   THE PROOF

The proof of the above theorem is by constructing a class of neural networks for which it is impossible to decide (for *all* instance) if a certain learning task can be satisfied. We will refer for this to "Hilbert's tenth problem" and show that for each of its instances we can construct a neural network, so that solutions to the loading problem would lead to solutions to the original problem (and vice versa). But as we know that Hilbert's tenth problem is unsolvable we also have to conclude that the loading problem we consider is unsolvable.

### 3.1   HILBERT'S TENTH PROBLEM

Our reference problem – of which we know it is unsolvable – is closely related to several famous and classical mathematical problems including for example Fermat's last theorem.

**Definition 1** *A* diophantine equation *is a polynomial $D$ in $n$ variables with integer coefficients, that is*

$$D(x_1, x_2, \ldots, x_n) = \sum_i d_i(x_1, x_2, \ldots, x_n)$$

*with each term $d_i$ of the form $d_i(x_1, x_2, \ldots, x_n) = c_i \cdot x_{i_1} \cdot x_{i_2} \cdots \cdots x_{i_m}$ where the indices $\{i_1, i_2, \ldots, i_m\}$ are taken from $\{1, 2, \ldots, n\}$ and the coefficient $c_i \in \mathbb{Z}$.*

The concrete problem, first formulated in [Hilbert, 1900] is to develop a universal algorithm how to find the *integer* solutions for all $D$, i.e. a vector $(x_1, x_2, \ldots, x_n)$ with $x_i \in \mathbb{Z}$ (or $\mathbb{N}$), such that $D(x_1, x_2, \ldots, x_n) = 0$. The corresponding decision problem therefore is the following:

**Decision 2 Hilbert's Tenth Problem**
*INSTANCE: Given a diophantine equation $D$.*
*QUESTION: Is there an integer solution for $D$?*

Although this problem might seem to be quite simple – it formulation is actually the shortest among D. Hilbert's famous 23 problems – it was not until 1970 when Y. Matijasevich could prove that it is *unsolvable* or *undecidable* [Matijasevich, 1970]. There is no recursive computable predicate for diophantine equations which holds if a solution in $\mathbb{Z}$ (or $\mathbb{N}$) exists and fails otherwise [Davis, 1973, Theorem 7.4].

## 3.2   THE NETWORK ARCHITECTURE

The construction of a neural network $N$ for each diophantine $D$ is now straight forward (see Fig.1). It is just a three step construction.

First, each variable $x_i$ of $D$ is represented in $N$ by a small sub-network. The structure of these modules is quite simple (left side of Fig.1). Note that only the self-recurrent connection for the unit at the bottom of these modules is "weighted" by $0.0 < w < 1.0$. All other connection transmit their signals unaltered (i.e. $w = 1.0$).

Second, the terms $d_i$ in $D$ are represent by $\Pi$-units in $N$ (as show in Fig.1). Therefore, the connections to these units from the sub-modules representing the variables $x_i$ of $D$ correspond to the occurrences of these variables in each term $d_i$.

Finally, the output signals of all these $\Pi$-units is multiplied by the corresponding coefficients $c_i$ and summed up by the $\Sigma$-unit at the top.

## 3.3   THE SUB-MODULES

The fundamental property of the networks constructed in the above way is given by the simple fact that the behavior of such a neural network $N$ corresponds uniquely to the evaluation of the original diophantine $D$.

First, note that the behavior of $N$ only depends on the weights $w_i$ in each of the variable modules. Therefore, we will take a closer look at the behavior of these sub-modules. Suppose, that at some initial moment a signal of value 1.0 is received by each variable module. After that the signal is reset again to 0.0.

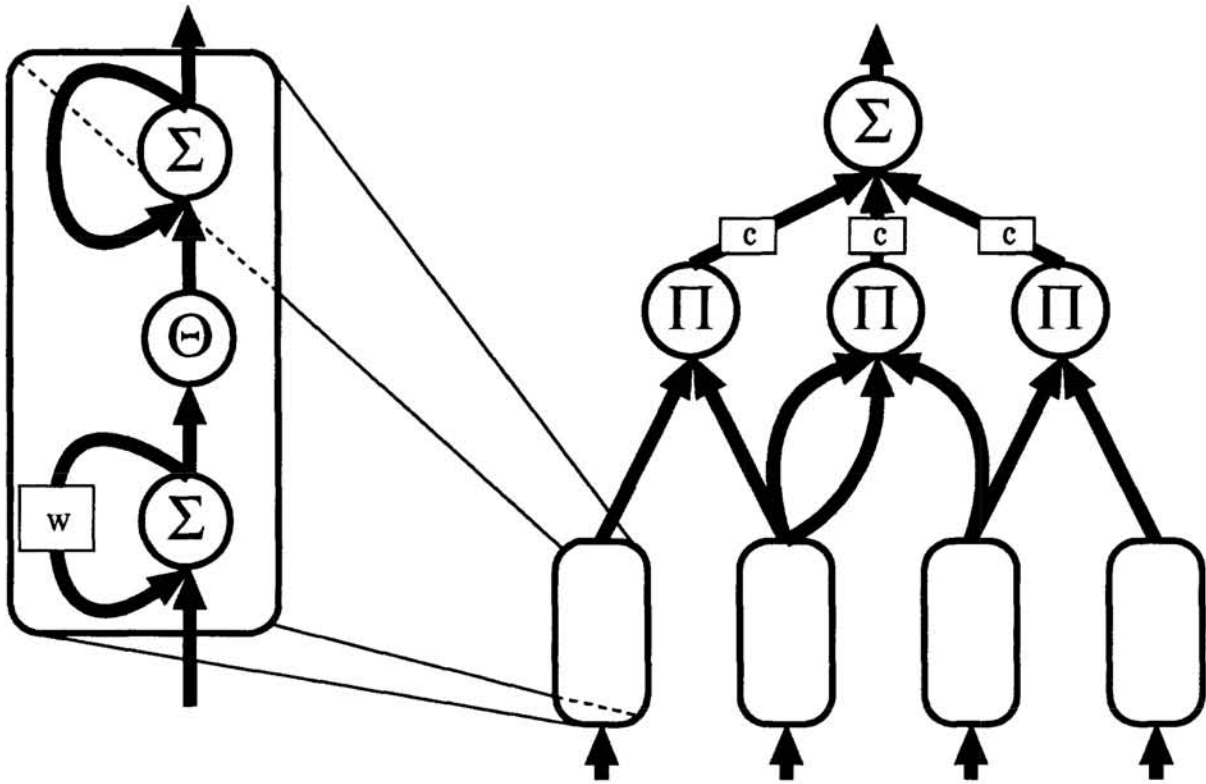

Figure 1: The network for $c_1 x_1 x_2 + c_2 x_2^2 x_3 + c_3 x_3 x_4$

The "seed" signal starts circling via $w_i$. With each update circle this signal becomes a little bit smaller. On the other hand, the same signal is also sent to the central $\Theta$-unit, which sends a signal 1.0 to the top accumulator unit as long as the "circling" activation of the bottom unit is larger then the (preset) threshold $\theta_i$. The top unit (which also keeps track of its former activiations via a recurrent connection) therefore just counts how many updates it takes before the activiation of the bottom unit drops below $\theta_i$.

The final, maximum, value which is emitted by the accumulator unit is some integer $x_i$ for which we have:

$$w_i^{x_i} > \theta_i > w_i^{x_i+1}$$

We thus have a correspondence between $w_i$ and the integer $x_i = \left\lfloor \frac{\ln \theta_i}{\ln w_i} \right\rfloor$, where $\lfloor x \rfloor$ the largest integer which is smaller or equal to $x$. Given $x_i$ we also can construct an appropriate weight $w_i$ by choosing it from the interval $\left( \exp\left( \frac{\ln \theta}{x_i} \right), \exp\left( \frac{\ln \theta}{x_i+1} \right) \right)$.

## 3.4   THE EQUIVALENCE

To conclude the proof, we now have to demonstrate the equivalence of Hilbert's tenth problem and the loading problem for the discussed class of recurrent networks and some learning task.

The learning task we will consider is the following: Map an input pattern with all signals equal to 1.0 (presented only once) to an output sequence which after a *finite* number of steps

is constant equal to 0.0. Note that – as discussed above – we could also consider a more static learing task where a final state, which determines the (single) output of the network, was determined by the condition that the outgoing signals of all $\Theta$-units had to be zero.

Considering this learing task and with what we said about the behavior of the sub-modules it is now trivial to see that the constructed network just evaluates the diophantine polynomial for a set of variables $x_i$ corresponding to the (final) output signals of the sub-modules (which are determined uniquely by the weight values $w_i$) if the input to the network is a pattern of all 1.0s.

If we had a solution $x_i$ of the original diophantine equation $D$, and if we take the corresponding values $w_i$ (according to the above relation) as weights in the sub-modules of $N$, then this would also solve the loading problem for this architecture. On the other hand, if we knew the correct weights $w_i$ for any such network $N$, then the corresponding integers $x_i$ would also solve the corresponding diophantine equation $D$.

In particular, if it would be possible to decide if a correct set of weights $w_i$ for $N$ exists (for the above learning task), we could also decide if the corresponding diophantine $D$ had a solution $x_i \in \mathbb{N}$ (and vice versa). As the whole construction was trivial, we have shown that both problems are equivalent.

# 4   CONCLUSIONS

We demonstrated that the loading problem not only is NP-complete – as shown for simple feed forward architectures in [Judd, 1990], [Lin and Vitter, 1991], [Blum and Rivest, 1992], etc. – but actually unsolvable, i.e. that the training of (recurrent) neural networks is among those problems which *"indeed are intractable in an especially strong sense"* [Garey and Johnson, 1979, p 12]. A related non-existence result concerning the training of higher order neural networks with integer weights was shown in [Wiklicky, 1992, Wiklicky, 1994].

One should stress once again that the fact that no general algorithm exists for higher order or recurrent networks, which could solve the loading problem (for all its instances), does not imply that *all* instances of this problem are unsolvable or that no solutions exist. One could hope, that in most relevant cases – whatever that could mean – or, when we restrict the problem, a sub-class of problems things might become tractable. But the difference between solvable and unsolvable problems often can be very small.

In particular, it is known that the problem of solving linear diophantine equations (instead of general ones) is polynomially computable, while if we go to quadratic diophantine equations the problem already becomes $NP$ complete [Johnson, 1990]. And for general diophantine the problem is even unsolvable. Moreover, it is also known that this problem is unsolvable if we consider only diophantine equations of maximum degree 4, and there exists a universal diophantine with only 13 variables which is unsolvable [Davis *et al.*, 1976].

But we think, that one should interpret the "negative" results on NP-complexity as well as on undecidability of the loading problem not as restrictions for neural networks, but as related to their computational power. As it was shown that concrete neural networks can be constructed, so that they simulate a universal Turing machine [Siegelmann and Sontag, 1992, Cosnard *et al.*, 1993]. It is mere the complexity of the problem one attempts to solve which simply cannot disappear and not some intrinsic intractability of the neural network approach.

## Acknowledgement

This work was started during the author's affiliation with the "Austrian Research Institute for Artificial Intelligence", Schottengasse 3, A-1010 Wien, Austria. Further work was supported by a grant from the Austrian "Fonds zur Förderung der wissenschaftlichen Forschung" as Projekt J0828-PHY.

## References

[Blum and Rivest, 1992] Avrim L. Blum and Ronald L. Rivest. Training a 3-node neural network is NP-complete. *Neural Networks*, 5:117–127, 1992.

[Cosnard *et al.*, 1993] Michael Cosnard, Max Garzon, and Pascal Koiran. Computability properties of low-dimensional dynamical systems. In *Symposium on Theoretical Aspects of Computer Science (STACS '93)*, pages 365–373, Springer-Verlag, Berlin – New York, 1993.

[Davis, 1973] Martin Davis. Hilbert's tenth problem is unsolvable. *Amer. Math. Monthly*, 80:233–269, March 1973.

[Davis *et al.*, 1976] Martin Davis, Yuri Matijasevich, and Julia Robinson. Hilbert's tenth problem - diophantine equations: Positive aspects of a negative solution. In Felix E. Browder, editor, *Mathematical developments arising from Hilbert*, pages 323–378, American Mathematical Society, 1976.

[Garey and Johnson, 1979] Michael R. Garey and David S. Johnson. *Computers and Intractability – A Guide to the Theory of NP-Completeness*. W. H. Freeman, New York, 1979.

[Hilbert, 1900] David Hilbert. Mathematische Probleme. *Nachr. Ges. Wiss. Göttingen, math.-phys.Kl.*, :253–297, 1900.

[Johnson, 1990] David S. Johnson. A catalog of complexity classes. In *Handbook of Theoretical Computer Science (Volume A: Algorithms and Complexity)*, chapter 2, pages 67–161, Elsevier – MIT Press, Amsterdam – Cambridge, Massachusetts, 1990.

[Judd, 1990] J. Stephen Judd. *Neural Network Design and the Complexity of Learning*. MIT Press, Cambridge, Massachusetts – London, England, 1990.

[Lin and Vitter, 1991] Jyh-Han Lin and Jeffrey Scott Vitter. Complexity results on learning by neural networks. *Machine Learning*, 6:211–230, 1991.

[Matijasevich, 1970] Yuri Matijasevich. Enumerable sets are diophantine. *Dokl. Acad. Nauk.*, 191:279–282, 1970.

[Siegelmann and Sontag, 1992] Hava T. Siegelmann and Eduardo D. Sontag. On the computational power of neural nets. In *Fifth Workshop on Computational Learning Theory (COLT '92)*, pages 440–449, 1992.

[Wiklicky, 1992] Herbert Wiklicky. *Synthesis and Analysis of Neural Networks — On a Framework for Artificial Neural Networks*. PhD thesis, University of Vienna – Technical University of Vienna, September 1992.

[Wiklicky, 1994] Herbert Wiklicky. The neural network loading problem is undecidable. In *Euro-COLT '93 – Conference on Computational Learning Theory*, page (to appear), Oxford University Press, Oxford, 1994.
